# Modeling Temporal Structure in Classical Conditioning

**Aaron C. Courville**[1,3] **and David S. Touretzky**[2,3]
[1]Robotics Institute, [2]Computer Science Department
[3]Center for the Neural Basis of Cognition
Carnegie Mellon University, Pittsburgh, PA 15213-3891
{*aaronc,dst*}*@cs.cmu.edu*

## Abstract

The Temporal Coding Hypothesis of Miller and colleagues [7] suggests that animals integrate related temporal patterns of stimuli into single memory representations. We formalize this concept using quasi-Bayes estimation to update the parameters of a constrained hidden Markov model. This approach allows us to account for some surprising temporal effects in the second order conditioning experiments of Miller et al. [1, 2, 3], which other models are unable to explain.

## 1 Introduction

Animal learning involves more than just predicting reinforcement. The well-known phenomena of latent learning and sensory preconditioning indicate that animals learn about stimuli in their environment before any reinforcement is supplied. More recently, a series of experiments by R. R. Miller and colleagues has demonstrated that in classical conditioning paradigms, animals appear to learn the temporal structure of the stimuli [8]. We will review three of these experiments. We then present a model of conditioning based on a constrained hidden Markov model, using quasi-Bayes estimation to adjust the model parameters online. Simulation results confirm that the model reproduces the experimental observations, suggesting that this approach is a viable alternative to earlier models of classical conditioning which cannot account for the Miller et al. experiments. Table 1 summarizes the experimental paradigms and the results.

**Expt. 1: Simultaneous Conditioning.** Responding to a conditioned stimulus (CS) is impaired when it is presented simultaneously with the unconditioned stimulus (US) rather than preceding the US. The failure of the simultaneous conditioning procedure to demonstrate a conditioned response (CR) is a well established result in the classical conditioning literature [9]. Barnet et al. [1] reported an interesting

|        | Phase 1 | Phase 2 | Test ⇒ Result | Test ⇒ Result |
|--------|---------|---------|---------------|---------------|
| Expt. 1 | (4)T+US | (4)C → T | T ⇒ − | C ⇒ CR |
| Expt. 2A | (12)T → C | (8)T → US | C ⇒ − | |
| Expt. 2B | (12)T → C | (8)T ⟶ US | C ⇒ CR | |
| Expt. 3 | (96)L → US → X | (8)B → X | X ⇒ − | B ⇒ CR |

Table 1: Experimental Paradigms. Phases 1 and 2 represent two stages of training trials, each presented $(n)$ times. The plus sign (+) indicates simultaneous presentation of stimuli; the short arrow (→) indicates one stimulus immediately following another; and the long arrow (⟶) indicates a 5 sec gap between stimulus offset and the following stimulus onset. For Expt. 1, the tone T, click train C, and footshock US were all of 5 sec duration. For Expt. 2, the tone and click train durations were 5 sec and the footshock US lasted 0.5 sec. For Expt. 3, the light L, buzzer B, and auditory stimulus X (either a tone or white noise) were all of 30 sec duration, while the footshock US lasted 1 sec. CR indicates a conditioned response to the test stimulus.

second-order extension of the classic paradigm. While a tone CS presented simultaneously with a footshock results in a minimal CR to the tone, a click train preceding the tone (in phase 2) does acquire associative strength, as indicated by a CR.

**Expt. 2: Sensory Preconditioning.** Cole et al. [2] exposed rats to a tone T immediately followed by a click train C. In a second phase, the tone was paired with a footshock US that either immediately followed tone offset (variant A), or occurred 5 sec after tone offset (variant B). They found that when C and US both immediately follow T, little conditioned response is elicited by the presentation of C. However, when the US occurs 5 sec after tone offset, so that it occurs later than C (measured relative to T), then C does come to elicit a CR.

**Expt. 3: Backward Conditioning.** In another experiment by Cole et al. [3], rats were presented with a flashing light L followed by a footshock US, followed by an auditory stimulus X (either a tone or white noise). In phase 2, a buzzer B was followed by X. Testing revealed that while X did not elicit a CR (in fact, it became a conditioned inhibitor), X did impart an excitatory association to B.

## 2    Existing Models of Classical Conditioning

The Rescorla-Wagner model [11] is still the best-known model of classical conditioning, but as a trial-level model, it cannot account for within-trial effects such as second order conditioning or sensitivity to stimulus timing. Sutton and Barto developed Y-dot theory [14] as a real-time extension of Rescorla-Wagner. Further refinements led to the Temporal Difference (TD) learning algorithm [14]. These extensions can produce second order conditioning. And using a memory buffer representation (what Sutton and Barto call a *complete serial compound*), TD can represent the temporal structure of a trial. However, TD cannot account for the empirical data in Experiments 1–3 because it does not make inferences about temporal relationships among stimuli; it focuses solely on predicting the US. In Experiment 1, some versions of TD can account for the reduced associative strength of a CS when its onset occurs simultaneously with the US, but no version of TD can explain why the second-order stimulus C should acquire greater associative strength than

T. In Experiment 2, no learning occurs in Phase 1 with TD because no prediction error is generated by pairing T with C. As a result, no CR is elicited by C after T has been paired with the US in Phase 2. In Experiment 3, TD fails to predict the results because X is not predictive of the US; thus X acquires no associative strength to pass on to B in the second phase.

Even models that predict future stimuli have trouble accounting for Miller et al.'s results. Dayan's "successor representation" [4], the world model of Sutton and Pinette [15], and the basal ganglia model of Suri and Schultz [13] all attempt to predict future stimulus vectors. Suri and Schultz's model can even produce one form of sensory preconditioning. However, none of these models can account for the responses in any of the three experiments in Table 1, because they do not make the necessary inferences about relations among stimuli.

**Temporal Coding Hypothesis** The temporal coding hypothesis (TCH) [7] posits that temporal contiguity is sufficient to produce an association between stimuli. A CS does not need to predict reward in order to acquire an association with the US. Furthermore, the association is not a simple scalar quantity. Instead, information about the temporal relationships among stimuli is encoded implicitly and automatically in the memory representation of the trial. Most importantly, TCH claims that memory representations of trials with similar stimuli become integrated in such a way as to preserve the relative temporal information [3].

If we apply the concept of memory integration to Experiment 1, we get the memory representation, C → T+US. If we interpret a CR as a prediction of *imminent* reinforcement, then we arrive at the correct prediction of a strong response to C and a weak response to T. Integrating the hypothesized memory representations of the two phases of Experiment 2 results in: A) T → C+US and B) T → C → US. The stimulus C is only predictive of the US in variant B, consistent with the experimental findings. For Experiment 3, an integrated memory representation of the two phases produces L+B → US → X. Stimulus B is predictive of the US while X is not. Thus, the temporal coding hypothesis is able to account for the results of each of the three experiments by associating stimuli with a timeline.

## 3   A Computational Model of Temporal Coding

A straightforward formalization of a timeline is a Markov chain of states. For this initial version of our model, state transitions within the chain are fixed and deterministic. Each state represents one instant of time, and at each timestep a transition is made to the next state in the chain. This restricted representation is key to capturing the phenomena underlying the empirical results. Multiple timelines (or Markov chains) emanate from a single holding state. The transitions out of this holding state are the only probabilistic and adaptive transitions in the simplified model. These transition probabilities represent the frequency with which the timelines are experienced. Figure 1 illustrates the model structure used in all simulations.

Our goal is to show that our model successfully integrates the timelines of the two training phases of each experiment. In the context of a collection of Markov chains, integrating timelines amounts to both phases of training becoming associated with a single Markov chain. Figure 1 shows the integration of the two phases of Expt. 2B.

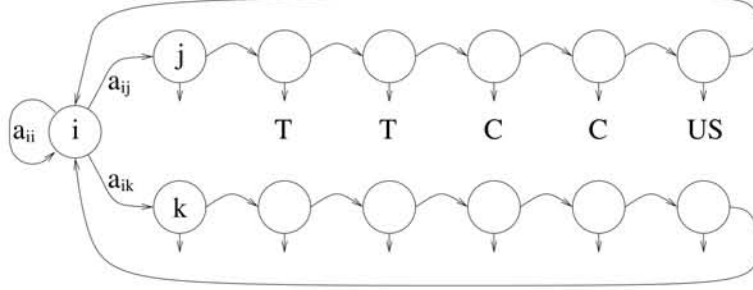

Figure 1: A depiction of the state and observation structure of the model. Shown are two timelines, one headed by state **j** and the other headed by state **k**. State **i**, the holding state, transitions to states **j** and **k** with probabilities $a_{ij}$ and $a_{ik}$ respectively. Below the timeline representations are a sequence of observations represented here as the symbols T, C and US. The T and C stimuli appear for two time steps each to simulate their presentation for an extended duration in the experiment.

During the second phase of the experiments, the second Markov chain (shown in Figure 1 starting with state **k**) offers an alternative to the chain associated with the first phase of learning. If we successfully integrate the timelines, this second chain is not used.

As suggested in Figure 1, associated with each state is a stimulus observation. "Stimulus space" is an $n$-dimensional continuous space, where $n$ is the number of distinct stimuli that can be observed (tone, light, shock, etc.) Each state has an expectation concerning the stimuli that should be observed when that state is occupied. This expectation is modeled by a probability density function, over this space, defined by a mixture of two multivariate Gaussians. The probability density at stimulus observation $\mathbf{x}^t$ in state $i$ at time $t$ is,

$$p(\mathbf{x}^t|s^t = i) = (1 - \omega_i) \cdot \mathcal{N}(\mathbf{x}^t|\mu_{i0}, \sigma_{i0}^2) + \omega_i \cdot \mathcal{N}(\mathbf{x}^t|\mu_{i1}, \sigma_{i1}^2), \tag{1}$$

where $\omega_i$ is a mixture coefficient for the two Gaussians associated with state $i$. The Gaussian means $\mu_{\mathbf{i0}}$ and $\mu_{\mathbf{i1}}$ and variances $\sigma_{\mathbf{i0}}^2$ and $\sigma_{\mathbf{i1}}^2$ are vectors of the same dimension as the stimulus vector $\mathbf{x^t}$. Given knowledge of the state, the stimulus components are assumed to be mutually independent (covariance terms are zero). We chose a continuous model of observations over a discrete observation model to capture stimulus generalization effects. These are not pursued in this paper.

For each state, the first Gaussian pdf is non-adaptive, meaning $\mu_{\mathbf{i0}}$ is fixed about a point in stimulus space representing the absence of stimuli. $\sigma_{\mathbf{i0}}^2$ is fixed as well. For the second Gaussian, $\mu_{\mathbf{i1}}$ and $\sigma_{\mathbf{i1}}^2$ are adaptive. This mixture of one fixed and one adaptive Gaussian is an approximation to the animal's belief distribution about stimuli, reflecting the observed tolerance animals have to absent expected stimuli. Put another way, animals seem to be less surprised by the absence of an expected stimulus than by the presence of an unexpected stimulus.

We assume that knowledge of the current state $s^t$ is inaccessible to the learner. This information must be inferred from the observed stimuli. In the case of a Markov chain, learning with hidden state is exactly the problem of parameter estimation in hidden Markov models. That is, we must update the estimates of $\omega$, $\mu_{\mathbf{1}}$ and $\sigma_{\mathbf{1}}^2$ for

each state, and $a_{ij}$ for each state transition (out of the holding state), in order to maximize the likelihood of the sequence of observations

The standard algorithm for hidden Markov model parameter estimation is the Baum-Welch method [10]. Baum-Welch is an off-line learning algorithm that requires all observations used in training to be held in memory. In a model of classical conditioning, this is an unrealistic assumption about animals' memory capabilities. We therefore require an online learning scheme for the hidden Markov model, with only limited memory requirements.

Recursive Bayesian inference is one possible online learning scheme. It offers the appealing property of combining prior beliefs about the world with current observations through the recursive application of Bayes' theorem, $p(\lambda|\mathbf{X}^t) \propto p(\mathbf{x}^t|\mathbf{X}^{t-1}, \lambda)p(\lambda|\mathbf{X}^{t-1})$. The prior distribution, $p(\lambda|\mathbf{X}^{t-1})$ reflects the belief over the parameter $\lambda$ before the observation at time $t$, $\mathbf{x}^t$. $\mathbf{X}^{t-1}$ is the observation history up to time $t-1$, i.e. $\mathbf{X}^{t-1} = \{\mathbf{x}^{t-1}, \mathbf{x}^{t-2}, \ldots\}$. The likelihood, $p(\mathbf{x}^t|\mathbf{X}^{t-1}, \lambda)$ is the probability density over $\mathbf{x}^t$ as a function of the parameter $\lambda$.

Unfortunately, the implementation of exact recursive Bayesian inference for a continuous density hidden Markov model (CDHMM) is computationally intractable. This is a consequence of there being missing data in the form of hidden state. With hidden state, the posterior distribution over the model parameters, after the observation, is given by

$$p(\lambda|\mathbf{X}^t) \propto \sum_{i=1}^{N} p(\mathbf{x}^t|s^t = i, \mathbf{X}^{t-1}, \lambda)p(s^t = i|\mathbf{X}^{t-1}, \lambda)p(\lambda|\mathbf{X}^{t-1}), \qquad (2)$$

where we have summed over the $N$ hidden states. Computing the recursion for multiple time steps results in an exponentially growing number of terms contributing to the exact posterior.

We instead use a recursive quasi-Bayes approximate inference scheme developed by Huo and Lee [5], who employ a quasi-Bayes approach [12]. The quasi-Bayes approach exploits the existence of a repeating distribution (natural conjugate) over the parameters for the *complete-data* CDHMM. (i.e. where missing data such as the state sequence is taken to be known). Briefly, we estimate the value of the missing data. We then use these estimates, together with the observations, to update the hyperparameters governing the prior distribution over the parameters (using Bayes' theorem). This results in an approximation to the exact posterior distribution over CDHMM parameters within the conjugate family of the complete-data CDHMM. See [5] for a more detailed description of the algorithm.

Estimating the missing data (hidden state) involves estimating transition probabilities between states, $\xi_{ij}^\tau = \Pr(s^\tau = i, s^{\tau+1} = j|\mathbf{X}^t, \lambda)$, and joint state and mixture component label probabilities $\zeta_{ik}^\tau = \Pr(s^\tau = i, l^\tau = k|\mathbf{X}^t, \lambda)$. Here $l^\tau = k$ is the mixture component label indicating which Gaussian, $k \in \{0, 1\}$, is the source of the stimulus observation at time $\tau$. $\lambda$ is the current estimate of all model parameters.

We use an online version of the forward-backward algorithm [6] to estimate $\xi_{ij}^\tau$ and $\zeta_{i1}^\tau$. The forward pass computes the joint probability over state occupancy (taken to be both the state value and the mixture component label) at time $\tau$ and the sequence of observations up to time $\tau$. The backward pass computes the probability of the observations in a memory buffer from time $\tau$ to the present time $t$ given the state

occupancy at time $\tau$. The forward and backward passes over state/observation sequences are combined to give an estimate of the state occupancy at time $\tau$ given the observations up to the present time $t$. In the simulations reported here the memory buffer was 7 time steps long ($t - \tau = 6$).

We use the estimates from the forward-backward algorithm together with the observations to update the hyperparameters. For the CDHMM, this prior is taken to be a product of Dirichlet probability density functions (pdfs) for the transition probabilities ($a_{ij}$), beta pdfs for the observation model mixture coefficients ($\omega_i$) and normal-gamma pdfs for the Gaussian parameters ($\mu_{i1}$ and $\sigma_{i1}^2$). The basic hyperparameters are exponentially weighted counts of events, with recency weighting determined by a forgetting parameter $\rho$. For example, $\kappa_{ij}$ is the number of expected transitions observed from state $i$ to state $j$, and is used to update the estimate of parameter $a_{ij}$. The hyperparameter $\nu_{ik}$ estimates the number of stimulus observations in state $i$ credited to Gaussian $k$, and is used to update the mixture parameter $\omega_i$. The remaining hyperparameters $\psi$, $\phi$, and $\theta$ serve to define the pdfs over $\mu_{i1}$ and $\sigma_{i1}^2$. The variable $d$ in the equations below indexes over stimulus dimensions. $S_{i1d}$ is an estimate of the sample variance, and is a constant in the present model.

$$\kappa_{ij}^\tau = \rho(\kappa_{ij}^{(\tau-1)} - 1) + 1 + \xi_{ij}^\tau \qquad \nu_{ik}^\tau = \rho(\nu_{ik}^{(\tau-1)} - 1) + 1 + \zeta_{ik}^\tau$$

$$\psi_{i1d}^\tau = \rho\psi_{i1d}^{(\tau-1)} + \zeta_{i1}^\tau \qquad \phi_{i1d}^\tau = \rho(\phi_{i1d}^{(\tau-1)} - \tfrac{1}{2}) + \tfrac{1+\zeta_{i1}^\tau}{2}$$

$$\theta_{i1d}^\tau = \rho\theta_{i1d}^{(\tau-1)} + \tfrac{\zeta_{i1}^\tau S_{i1d}}{2} + \tfrac{\rho\psi_{i1d}^{(\tau-1)}\zeta_{i1}^\tau}{2(\rho\psi_{i1d}^{(\tau-1)}+\zeta_{i1}^\tau)}(x_d^\tau - \mu_{i1d}^{(\tau-1)})^2$$

In the last step of our inference procedure, we update our estimate of the model parameters as the mode of their approximate posterior distribution. While this is an approximation to proper Bayesian inference on the parameter values, the mode of the approximate posterior is guaranteed to converge to a mode of the exact posterior. In the equations below, $N$ is the number of states in the model.

$$a_{ij}^\tau = \frac{\kappa_{ij}^\tau - 1}{\sum_{n=1}^N (\kappa_{in}^\tau - 1)} \qquad \omega_i^\tau = \frac{\nu_{i1}^\tau - 1}{\nu_{i0}^\tau + \nu_{i1}^\tau - 2} \qquad \mu_{i1d}^\tau = \frac{\rho\psi_{i1d}^{(\tau-1)}\mu_{i1d}^{(\tau-1)} + \zeta_{i1}^\tau x_d^\tau}{\rho\psi_{i1d}^{(\tau-1)} + \zeta_{i1}^\tau}$$

$$(\sigma_{i1d}^2)^\tau = \frac{2\rho\theta_{i1d}^{(\tau-1)} + \rho\psi_{i1d}^{(\tau-1)}\cdot(\mu_{i1d}^\tau - \mu_{i1d}^{(\tau-1)})^2 + \zeta_{i1}^\tau S_{i1d} + \zeta_{i1}^\tau(x_d^\tau - \mu_{i1d}^\tau)^2}{\rho(2\phi_{i1d}^{(\tau-1)} - 1) + \zeta_{i1}^\tau}$$

## 4   Results and Discussion

The model contained two timelines (Markov chains). Let $i$ denote the holding state and $j, k$ the initial states of the two chains. The transition probabilities were initialized as $a_{ij} = a_{ik} = 0.025$ and $a_{ii} = 0.95$. Adaptive Gaussian means $\mu_{i1d}$ were initialized to small random values around a baseline of $10^{-4}$ for all states. The exponential forgetting factor was $\rho = 0.9975$, and both the sample variances $S_{i1d}$ and the fixed variances $\sigma_{i0d}^2$ were set to 0.2.

We trained the model on each of the experimental protocols of Table 1, using the same numbers of trials reported in the original papers. The model was run continuously through both phases of the experiments with a random intertrial interval.

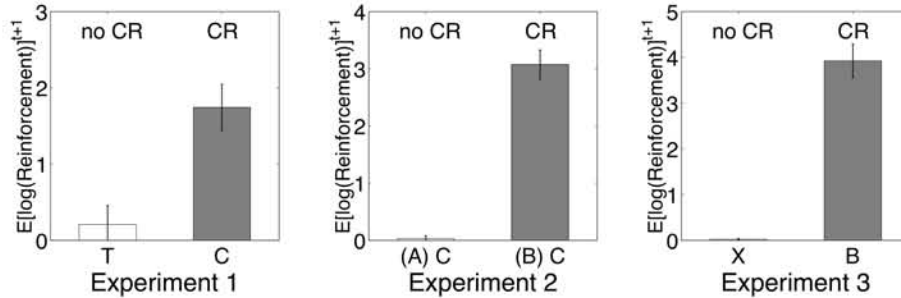

Figure 2: Results from 20 runs of the model simulation with each experimental paradigm. On the ordinate is the total reinforcement (US), on a log scale, above the baseline (an arbitrary perception threshold) expected to occur on the next time step. The error bars represent two standard deviations away from the mean.

Figure 2 shows the simulation results from each of the three experiments. If we assume that the CR varies monotonically with the US prediction, then in each case, the model's predicted CR agreed with the observations of Miller et al.

The CR predictions are the result of the model integrating the two phases of learning into one timeline. At the time of the presentation of the Phase 2 stimuli, the states forming the timeline describing the Phase 1 pattern of stimuli were judged more likely to have produced the Phase 2 stimuli than states in the other timeline, which served as a null hypothesis. In another experiment, not shown here, we trained the model on disjoint stimuli in the two phases. In that situation it correctly chose a separate timeline for each phase, rather than merging the two.

We have shown that under the assumption that observation probabilities are modeled by a mixture of Gaussians, and a very restrictive state transition structure, a hidden Markov model can integrate the memory representations of similar temporal stimulus patterns. "Similarity" is formalized in this framework as likelihood under the timeline model. We propose this model as a mechanism for the integration of memory representations postulated in the Temporal Coding Hypothesis.

The model can be extended in many ways. The current version assumes that event chains are long enough to represent an entire trial, but short enough that the model will return to the holding state before the start of the next trial. An obvious refinement would be a mechanism to dynamically adjust chain lengths based on experience. We are also exploring a generalization of the model to the semi-Markov domain, where state occupancy duration is modeled explicitly as a pdf. State transitions would then be tied to changes in observations, rather than following a rigid progression as is currently the case. Finally, we are experimenting with mechanisms that allow new chains to be split off from old ones when the model determines that current stimuli differ consistently from the closest matching timeline.

Fitting stimuli into existing timelines serves to maximize the likelihood of current observations in light of past experience. But why should animals learn the temporal structure of stimuli as timelines? A collection of timelines may be a reasonable model of the natural world. If this is true, then learning with such a strong inductive bias may help the animal to bring experience of related phenomena to bear in novel situations—a desirable characteristic for an adaptive system in a changing world.

## Acknowledgments

Thanks to Nathaniel Daw and Ralph Miller for helpful discussions. This research was funded by National Science Foundation grants IRI-9720350 and IIS-9978403. Aaron Courville was funded in part by a Canadian NSERC PGS B fellowship.

## References

[1] R. C. Barnet, H. M. Arnold, and R. R. Miller. Simultaneous conditioning demonstrated in second-order conditioning: Evidence for similar associative structure in forward and simultaneous conditioning. *Learning and Motivation*, 22:253–268, 1991.

[2] R. P. Cole, R. C. Barnet, and R. R. Miller. Temporal encoding in trace conditioning. *Animal Learning and Behavior*, 23(2):144–153, 1995.

[3] R. P. Cole and R. R. Miller. Conditioned excitation and conditioned inhibition acquired through backward conditioning. *Learning and Motivation*, 30:129–156, 1999.

[4] P. Dayan. Improving generalization for temporal difference learning: the successor representation. *Neural Computation*, 5:613–624, 1993.

[5] Q. Huo and C.-H. Lee. On-line adaptive learning of the continuous density hidden Markov model based on approximate recursive Bayes estimate. *IEEE Transactions on Speech and Audio Processing*, 5(2):161–172, 1997.

[6] V. Krishnamurthy and J. B. Moore. On-line estimation of hidden Markov model parameters based on the Kullback-Leibler information measure. *IEEE Transactions on Signal Processing*, 41(8):2557–2573, 1993.

[7] L. D. Matzel, F. P. Held, and R. R. Miller. Information and the expression of simultaneous and backward associations: Implications for contiguity theory. *Learning and Motivation*, 19:317–344, 1988.

[8] R. R. Miller and R. C. Barnet. The role of time in elementary associations. *Current Directions in Psychological Science*, 2(4):106–111, 1993.

[9] I. P. Pavlov. *Conditioned Reflexes*. Oxford University Press, 1927.

[10] L. R. Rabiner. A tutorial on hidden Markov models and selected applications in speech recognition. *Proceedings of the IEEE*, 77(2):257–285, 1989.

[11] R. A. Rescorla and A. R. Wagner. A theory of Pavlovian conditioning: Variations in the effectiveness of reinforcement and nonreinforcement. In A. H. Black and W. F. Prokasy, editors, *Classical Conditioning II*. Appleton-Century-Crofts, 1972.

[12] A. F. M. Smith and U. E. Makov. A quasi-Bayes sequential procedure for mixtures. *Journal of the Royal Statistical Society*, 40(1):106–112, 1978.

[13] R. E. Suri and W. Schultz. Temporal difference model reproduces anticipatory neural activity. *Neural Computation*, 13(4):841–862, 2001.

[14] R. S. Sutton and A. G. Barto. Time-derivative models of Pavlovian reinforcement. In M. Gabriel and J. Moore, editors, *Learning and Computational Neuroscience: Foundations of Adaptive Networks*, chapter 12, pages 497–537. MIT Press, 1990.

[15] R. S. Sutton and B. Pinette. The learning of world models by connectionist networks. In L. Erlbaum, editor, *Proceedings of the seventh annual conference of the cognitive science society*, pages 54–64, Irvine, California, August 1985.